# A Sequence Kernel and its Application to Speaker Recognition

**William M. Campbell**
Motorola Human Interface Lab
7700 S. River Parkway
Tempe, AZ 85284
*Bill.Campbell@motorola.com*

## Abstract

A novel approach for comparing sequences of observations using an explicit-expansion kernel is demonstrated. The kernel is derived using the assumption of the independence of the sequence of observations and a mean-squared error training criterion. The use of an explicit expansion kernel reduces classifier model size and computation dramatically, resulting in model sizes and computation one-hundred times smaller in our application. The explicit expansion also preserves the computational advantages of an earlier architecture based on mean-squared error training. Training using standard support vector machine methodology gives accuracy that significantly exceeds the performance of state-of-the-art mean-squared error training for a speaker recognition task.

## 1   Introduction

Comparison of sequences of observations is a natural and necessary operation in speech applications. Several recent approaches using support vector machines (SVM's) have been proposed in the literature. The first set of approaches attempts to model emission probabilities for hidden Markov models [1, 2]. This approach has been moderately successful in reducing error rates, but suffers from several problems. First, large training sets result in long training times for support vector methods. Second, the emission probabilities must be approximated [3], since the output of the support vector machine is not a probability. A more recent method for comparing sequences is based on the Fisher kernel proposed by Jaakkola and Haussler [4]. This approach has been explored for speech recognition in [5]. The application to speaker recognition is detailed in [6]. We propose an alternative kernel based upon polynomial classifiers and the associated mean-squared error (MSE) training criterion [7]. The advantage of this kernel is that it preserves the structure of the classifier in [7] which is both computationally and memory efficient.

We consider the application of *text-independent* speaker recognition; i.e., determining or verifying the identity of an individual through voice characteristics. Text-independent recognition implies that knowledge of the text of the speech data is not used. Traditional methods for text-independent speaker recognition are vector quantization [8], Gaussian mixture models [9], and artificial neural networks [8]. A state-of-the-art approach based on polynomial classifiers was presented in [7]. The polynomial approach has several ad-

vantages over traditional methods–1) it is extremely computationally-efficient for identification, 2) the classifier is discriminative which eliminates the need for a background or cohort model [10], and 3) the method generates small classifier models.

In Section 2, we describe polynomial classifiers and the associated scoring process. In Section 3, we review the process for mean-squared error training. Section 4 introduces the new kernel. Section 5 compares the new kernel approach to the standard mean-squared error training approach.

## 2   Polynomial classifiers for sequence data

We start by considering the problem of speaker verification–a two-class problem. In this case, the goal is to determine the correctness of an identity claim (e.g., a user id was entered in the system) from a voice input. If $\omega$ is the class, then the decision to be made is if the claim is valid, $\omega = \text{spk}$, or if an impostor is trying to break into the system, $\omega = \text{imp}$. We motivate the classification process from a probabilistic viewpoint.

For the verification application, a decision is made from a sequence of observations extracted $\mathbf{x}_1, \ldots, \mathbf{x}_n$ from the speech input. We decide based on the output of a discriminant function using a polynomial classifier. A polynomial classifier of the form $f(\mathbf{x}) = \mathbf{w}^t \mathbf{p}(\mathbf{x})$ where $\mathbf{w}$ is the vector of classifier parameters (model) and $\mathbf{p}$ is an expansion of the input space into the vector of monomials of degree $K$ or less is used. For example, if $K = 3$ and $\mathbf{x} = \begin{bmatrix} x_1 & x_2 \end{bmatrix}^t$, then

$$\mathbf{p}(\mathbf{x}) = \begin{bmatrix} 1 & x_1 & x_2 & x_1^2 & x_1 x_2 & x_2^2 & x_1^3 & x_1^2 x_2 & x_1 x_2^2 & x_2^3 \end{bmatrix}^t. \tag{1}$$

Note that we do not use a nonlinear activation function as is common in higher-order neural networks; this allows us to find a closed form solution for training. Also, note that we use a bold $\mathbf{p}$ to avoid confusion with probabilities.

If the polynomial classifier is trained with a mean-squared error training criterion and target values of 1 for $\omega = \text{spk}$ and 0 for $\omega = \text{imp}$, then $f(\mathbf{x})$ will approximate the *a posteriori* probability $p(\omega = \text{spk}|\mathbf{x})$ [11]. We can then find the probability of the entire sequence, $p(\mathbf{x}_1, \ldots, \mathbf{x}_n | \omega = \text{spk})$, as follows. Assuming independence of the observations [12] gives

$$
\begin{aligned}
p(\mathbf{x}_1, \ldots, \mathbf{x}_n | \omega) &= \prod_{i=1}^{n} p(\mathbf{x}_i | \omega) \\
&= \prod_{i=1}^{n} \frac{p(\omega | \mathbf{x}_i) p(\mathbf{x}_i)}{p(\omega)}.
\end{aligned}
\tag{2}
$$

For the purposes of classification, we can discard $p(\mathbf{x}_i)$. We take the logarithm of both sides to get the discriminant function

$$d'(\mathbf{x}_1^n | \omega) = \sum_{i=1}^{n} \log \left( \frac{p(\omega | \mathbf{x}_i)}{p(\omega)} \right) \tag{3}$$

where we have used the shorthand $\mathbf{x}_1^n$ to denote the sequence $\mathbf{x}_1, \ldots, \mathbf{x}_n$. We use two terms of the Taylor series, $\log(x) \approx x - 1$, to approximate the discriminant function and also normalize by the number of frames to obtain the final discriminant function

$$d(\mathbf{x}_1^n | \omega) = \frac{1}{n} \sum_{i=1}^{n} \frac{p(\omega | \mathbf{x}_i)}{p(\omega)}. \tag{4}$$

Note that we have discarded the $-1$ in this discriminant function since this will not affect the classification decision. The key reason for using the Taylor approximation is that it reduces computation without significantly affecting classifier accuracy.

Now assume we have a polynomial function $f(\mathbf{x}) \approx p(\omega = \text{spk}|\mathbf{x})$; we call the vector $\mathbf{w}$ the speaker model. Substituting in the polynomial function $f(\mathbf{x})$ gives

$$
\begin{aligned}
d(\mathbf{x}_1^n | \omega = \text{spk}) &= \frac{1}{n} \sum_{i=1}^{n} \frac{\mathbf{w}^t \mathbf{p}(\mathbf{x}_i)}{p(\omega = \text{spk})} \\
&= \frac{1}{np(\omega = \text{spk})} \mathbf{w}^t \left( \sum_{i=1}^{n} \mathbf{p}(\mathbf{x}_i) \right) \\
&= \frac{1}{p(\omega = \text{spk})} \mathbf{w}^t \bar{\mathbf{p}}
\end{aligned}
\tag{5}
$$

where we have defined the mapping $\mathbf{x}_1^n \to \bar{\mathbf{p}}$ as

$$
\mathbf{x}_1^n \to \frac{1}{n} \sum_{i=1}^{n} \mathbf{p}(\mathbf{x}_i).
\tag{6}
$$

We summarize the scoring method. For a sequence of input vectors $\mathbf{x}_1, \ldots \mathbf{x}_n$ and a speaker model, $\mathbf{w}$, we construct $\bar{\mathbf{p}}$ using (6). We then score using the speaker model, $\text{score} = \mathbf{w}^t \bar{\mathbf{p}}$. Since we are performing verification, if score is above a threshold then we declare the identity claim valid; otherwise, the claim is rejected as an impostor attempt. More details on this probabilistic scoring method can be found in [13].

Extending the sequence scoring framework to the case of identification (i.e., identifying the speaker from a list of speakers by voice) is straightforward. In this case, we construct speaker models for each speaker $\mathbf{w}_i$ and then choose the speaker $i$ which maximizes $\mathbf{w}_i^t \bar{\mathbf{p}}$ (assuming equal prior probability of each speaker). Note that identification has low computational complexity, since we must only compute one inner product to determine the speaker's score.

## 3 Mean-squared error training

We next review how to train the polynomial classifier to approximate the probability $p(\omega|\mathbf{x})$; this process will help us set notation for the following sections. Let $\mathbf{w}$ be the desired speaker model and $y(\omega)$ the ideal output; i.e., $y(\text{spk}) = 1$ and $y(\text{imp}) = 0$. The resulting problem is

$$
\mathbf{w}^* = \underset{\mathbf{w}}{\text{argmin}} \, \mathbf{E} \left\{ \left( \mathbf{w}^t \mathbf{p}(\mathbf{x}) - y(\omega) \right)^2 \right\}
\tag{7}
$$

where $\mathbf{E}$ denotes expectation. This criterion can be approximated using the training set as

$$
\mathbf{w}^* = \underset{\mathbf{w}}{\text{argmin}} \left[ \sum_{i=1}^{N_{\text{spk}}} \left| \mathbf{w}^t \mathbf{p}(\mathbf{x}_i) - 1 \right|^2 + \sum_{i=1}^{N_{\text{imp}}} \left| \mathbf{w}^t \mathbf{p}(\mathbf{y}_i) \right|^2 \right].
\tag{8}
$$

Here, the speaker's training data is $\mathbf{x}_1, \ldots, \mathbf{x}_{N_{\text{spk}}}$, and the anti-speaker data is $\mathbf{y}_1, \ldots, \mathbf{y}_{N_{\text{imp}}}$. (Anti-speakers are designed to have the same statistical characteristics as the impostor set.)

The training method can be written in matrix form. First, define $\mathbf{M}_{\text{spk}}$ as the matrix whose rows are the polynomial expansion of the speaker's data; i.e.,

$$
\mathbf{M}_{\text{spk}} = \begin{bmatrix} \mathbf{p}(\mathbf{x}_1)^t \\ \mathbf{p}(\mathbf{x}_2)^t \\ \vdots \\ \mathbf{p}(\mathbf{x}_{N_{\text{spk}}})^t \end{bmatrix}.
\tag{9}
$$

Define a similar matrix for the impostor data, $\mathbf{M}_{\mathrm{imp}}$. Define

$$\mathbf{M} = \begin{bmatrix} \mathbf{M}_{\mathrm{spk}} \\ \mathbf{M}_{\mathrm{imp}} \end{bmatrix}. \tag{10}$$

The problem (8) then becomes

$$\mathbf{w}^* = \underset{\mathbf{w}}{\mathrm{argmin}} \, \|\mathbf{Mw} - \mathbf{o}\|_2 \tag{11}$$

where $\mathbf{o}$ is the vector consisting of $N_{\mathrm{spk}}$ ones followed by $N_{\mathrm{imp}}$ zeros (i.e., the ideal output).

The problem (11) can be solved using the method of normal equations,

$$\mathbf{M}^t \mathbf{M} \mathbf{w} = \mathbf{M}^t \mathbf{o}. \tag{12}$$

We rearrange (12) to

$$\left(\mathbf{M}^t \mathbf{M}\right) \mathbf{w} = \mathbf{M}_{\mathrm{spk}}^t \mathbf{1} \tag{13}$$

where $\mathbf{1}$ is the vector of all ones. If we define $\mathbf{R} = \mathbf{M}^t \mathbf{M}$ and solve for $\mathbf{w}$, then (13) becomes

$$\mathbf{w} = \mathbf{R}^{-1} \mathbf{M}_{\mathrm{spk}}^t \mathbf{1}. \tag{14}$$

## 4    The naive a posteriori sequence kernel

We can now combine the methods from Sections 2 and 3 to obtain a novel sequence comparison kernel in a straightforward manner. Combine the speaker model from (14) with the scoring equation from (5) to obtain the classifier score

$$\mathrm{score} = \frac{1}{p(\omega = \mathrm{spk})} \bar{\mathbf{p}}^t \mathbf{w} = \frac{1}{p(\omega = \mathrm{spk})} \bar{\mathbf{p}}^t \mathbf{R}^{-1} \mathbf{M}_{\mathrm{spk}}^t \mathbf{1}. \tag{15}$$

Now $p(\omega = \mathrm{spk}) = N_{\mathrm{spk}}/(N_{\mathrm{imp}} + N_{\mathrm{spk}}) \approx N_{\mathrm{spk}}/N_{\mathrm{imp}}$ (because of the large anti-speaker population), so that (15) becomes

$$\mathrm{score} = \bar{\mathbf{p}}^t \bar{\mathbf{R}}^{-1} \bar{\mathbf{p}}_{\mathrm{train}}^t \tag{16}$$

where $\bar{\mathbf{p}}_{\mathrm{train}}$ is $(1/N_{\mathrm{spk}}) \mathbf{M}_{\mathrm{spk}}^t \mathbf{1}$ (note that this exactly the same as mapping the training data using (6)), and $\bar{\mathbf{R}}$ is $(1/N_{\mathrm{imp}}) \mathbf{R}$.

The scoring method in (16) is the basis of our sequence kernel. Given two sequences of speech feature vectors, $\mathbf{x}_1^n$ and $\mathbf{y}_1^m$, we compare them by mapping $\mathbf{x}_1^n \to \bar{\mathbf{p}}_x$ and $\mathbf{y}_1^m \to \bar{\mathbf{p}}_y$ and then computing

$$k_{\mathrm{naps}}(\mathbf{x}_1^n, \mathbf{y}_1^m) = \bar{\mathbf{p}}_x^t \bar{\mathbf{R}}^{-1} \bar{\mathbf{p}}_y. \tag{17}$$

We call $k_{\mathrm{naps}}$ the **n**aive **a p**osteriori **s**equence kernel since scoring assumes independence of observations and training approximates the a posteriori probabilities. The value $k_{\mathrm{naps}}(\mathbf{x}_1^n, \mathbf{y}_1^m)$ can be interpreted as scoring using a polynomial classifier on the sequence $\mathbf{y}_1^m$, see (5), with the MSE model trained from feature vectors $\mathbf{x}_1^m$ (or vice-versa because of symmetry).

Several observations should be made about the NAPS kernel. First, scoring complexity can be reduced dramatically in training by the following trick. We factor $\bar{\mathbf{R}}^{-1} = \bar{\mathbf{U}}^t \mathbf{U}$ using the Cholesky decomposition. Then $k_{\mathrm{naps}}(\mathbf{x}_1^n, \mathbf{y}_1^m) = (\mathbf{U}\bar{\mathbf{p}}_x)^t (\mathbf{U}\bar{\mathbf{p}}_y)$. I.e., if we transform all the sequence data by $\mathbf{U}\bar{\mathbf{p}}_x$ before training, the sequence kernel is a simple inner product. For our application in Section 5, this reduces training time from 5 hours per speaker down to 40 seconds on a Sun Ultra 60, 360 MHz. Second, since the NAPS kernel explicitly performs the expansion to "feature space", we can simplify the output of the support vector machine. Suppose $g(\bar{\mathbf{p}})$ is the (soft) output of the SVM,

$$g(\bar{\mathbf{p}}) = \sum_{i=1}^{l} \alpha_i y_i k_{\mathrm{naps}}(\bar{\mathbf{p}}_i, \bar{\mathbf{p}}) + b. \tag{18}$$

We can simplify this to

$$g(\bar{\mathbf{p}}) = \left( \sum_{i=1}^{l} \alpha_i y_i \bar{\mathbf{R}}^{-1} \bar{\mathbf{p}}_i + \mathbf{b} \right)^t \bar{\mathbf{p}} \tag{19}$$

where $\mathbf{b} = \begin{bmatrix} b & 0 & \ldots & 0 \end{bmatrix}^t$. That is, once we train the support vector machine, we can collapse all the support vectors down into a single model $\mathbf{w}$, where $\mathbf{w}$ is the quantity in parenthesis in (19). Third, although the NAPS kernel is reminiscent of the Mahalanobis distance, it is distinct. No assumption of equal covariance matrices for different classes (speakers) is made for the new kernel–the kernel covariance matrix is a mixture of the individual class covariances. Also, the kernel is not a distance measure–no subtraction of means occurs as in the Mahalanobis distance.

## 5 Results

### 5.1 Setup

The NAPS kernel was tested on the standard speaker recognition database YOHO [14] collected from 138 speakers. Utterances in the database consist of combination lock phrases of fixed length; e.g., "23-45-56." Enrollment and verification session were recorded at distinct times. (Enrollment is the process of collecting data for training and generating a speaker model. Verification is the process of testing the system; i.e., the user makes an identity claim and then this hypothesis is verified.) For each speaker, enrollment consisted of four sessions each containing twenty-four utterances. Verification consisted of ten separate sessions with four utterances per session (again per speaker). Thus, there are 40 tests of the speaker's identity and 40*137=5480 possible impostor attempts on a speaker. For clarity, we emphasize that enrollment and verification session data is completely separate.

To extract features for each of the utterances, we used standard speech processing. Each utterance was broken up into frames of 30 ms each with a frame rate of 100 frames/sec. The mean was removed from each frame, and the frame was preemphasized with the filter $1 - 0.97z^{-1}$. A Hamming window was applied and then 12 linear prediction coefficients were found. The resulting coefficients were transformed to 12 cepstral coefficients. Endpointing was performed to eliminate non-speech frames. This typically resulted in approximately 200 observations per utterance.

For verification, we measure performance in terms of the pooled and average equal error rates (EER). The average EER is found by averaging the individual EER for each speaker. The individual EER is the threshold at which the false accept rate (FAR) equals the false reject rate (FRR). The pooled EER is found by setting a constant threshold across the entire population. When the FAR equals the FRR for the entire population this is termed the pooled EER. For identification, the misclassification error rate is used.

To eliminate bias in verification, we trained the first 69 speakers against the first 69 and the second 69 against the second 69 (as in [7]). We then performed verification using the second 69 as impostors to the first 69 speakers models and vice versa. This insures that the impostors are unknown. For identification, we trained all 138 speakers against each other.

### 5.2 Experiments

We trained support vector machines for each speaker using the software tool SVM-Torch [15] and the NAPS kernel (17). The 12 cepstral features were mapped to a dimension 455 vector using a 3rd degree polynomial classifier. Single utterances (i.e., "23-45-56") were converted to single vectors using the mapping (6) and then transformed with

the Cholesky factor to reduce computation. We cross-validated using the first 3 enrollment sessions as training and the 4th enrollment session as a test to determine the best tradeoff between margin and error; the best performing value of $c = 0.1$ was used with the final SVMTorch training. Using the identical set of features and the same methodology, classifier models were also trained using the mean-squared error criterion using the method in [7]. For final testing, all 4 enrollment session were used for training, and all verification sessions were used for testing.

Results for verification and identification are shown in Table 1. The new kernel method reduces error rates considerably–the average EER is reduced by $38\%$, the pooled EER is reduced by $47\%$, and the identification error rate is reduced by $42\%$. The average number of support vectors was $139$ which resulted in a model size of about $253,540$ bytes (in single precision floating point); using the model size reduction method in Section 4 resulted in a model size of 1820 bytes–over a hundred times reduction in size.

Table 1: Comparison of structural risk minimization and MSE training

|  | MSE | NAPS SVM |
|---|---|---|
| Average EER | 1.63% | 1.01% |
| Pooled EER | 2.76% | 1.45% |
| ID error rate | 4.71% | 2.72% |

We also plotted scores for all speakers versus a threshold, see Figure 1. We normalized the scores for the MSE and SVM approaches to the same range for comparison. One can easily see the reduction in pooled EER from the graph. Note also the dramatic shifting of the FRR curve to the right for the SVM training, resulting in substantially better error rates than the MSE training. For instance, when FAR is $0.1\%$, the MSE training method gives an FRR of $45\%$; whereas, the SVM training method gives an FRR of $12\%$–a reduction by a factor of $3.75$ in error.

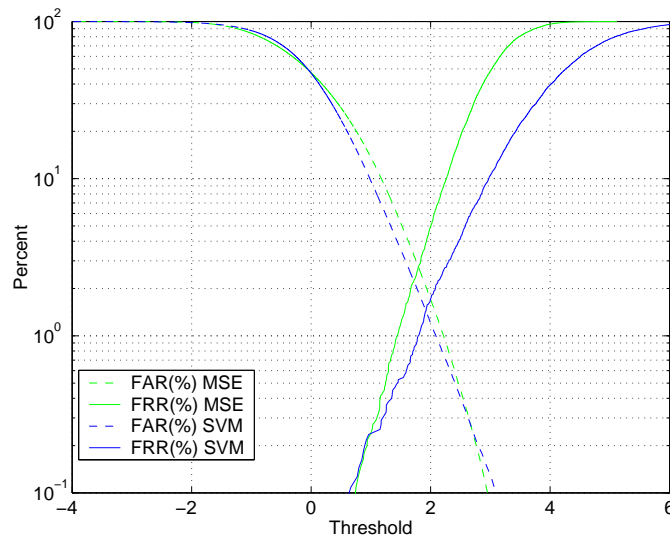

Figure 1: FAR/FRR rates for the entire population versus a threshold for the SVM and MSE training methods

# 6 Conclusions and future work

A novel kernel for comparing sequences in speech applications was derived, the NAPS kernel. This data-dependent kernel was motivated by using a probabilistic scoring method and mean-squared error training. Experiments showed that incorporating this kernel in an SVM training architecture yielded performance superior to that of the MSE training criterion. Reduction in error rates of up to $3.75$ times were observed while retaining the efficiency of the original MSE classifier architecture.

The new kernel method is also applicable to more general situations. Potential applications include–using the approach with radial basis functions, application to automatic speech recognition, and extending to an SVM/HMM architecture.

# References

[1] Vincent Wan and William M. Campbell, "Support vector machines for verification and identification," in *Neural Networks for Signal Processing X, Proceedings of the 2000 IEEE Signal Processing Workshop*, 2000, pp. 775–784.

[2] Aravind Ganapathiraju and Joseph Picone, "Hybrid SVM/HMM architectures for speech recognition," in *Speech Transcription Workshop*, 2000.

[3] John C. Platt, "Probabilities for SV machines," in *Advances in Large Margin Classifiers*, Alexander J. Smola, Peter L. Bartlett, Bernhard Schölkopf, and Dale Schuurmans, Eds., pp. 61–74. The MIT Press, 2000.

[4] Tommi S. Jaakkola and David Haussler, "Exploiting generative models in discriminative classifiers," in *Advances in Neural Information Processing 11*, M. S. Kearns, S. A. Solla, and D. A. Cohn, Eds. 1998, pp. 487–493, The MIT Press.

[5] Nathan Smith, Mark Gales, and Mahesan Niranjan, "Data-dependent kernels in SVM classification of speech patterns," Tech. Rep. CUED/F-INFENG/TR.387, Cambridge University Engineering Department, 2001.

[6] Shai Fine, Jiří Navrátil, and Ramesh A. Gopinath, "A hybrid GMM/SVM approach to speaker recognition," in *Proceedings of the International Conference on Acoustics, Speech, and Signal Processing*, 2001.

[7] William M. Campbell and Khaled T. Assaleh, "Polynomial classifier techniques for speaker verification," in *Proceedings of the International Conference on Acoustics, Speech, and Signal Processing*, 1999, pp. 321–324.

[8] Kevin R. Farrell, Richard J. Mammone, and Khaled T. Assaleh, "Speaker recognition using neural networks and conventional classifiers," *IEEE Trans. on Speech and Audio Processing*, vol. 2, no. 1, pp. 194–205, Jan. 1994.

[9] Douglas A. Reynolds, "Automatic speaker recognition using Gaussian mixture speaker models," *The Lincoln Laboratory Journal*, vol. 8, no. 2, pp. 173–192, 1995.

[10] Michael J. Carey, Eluned S. Parris, and John S. Bridle, "A speaker verification system using alpha-nets," in *Proceedings of the International Conference on Acoustics Speech and Signal Processing*, 1991, pp. 397–400.

[11] Jürgen Schürmann, *Pattern Classification*, John Wiley and Sons, Inc., 1996.

[12] Lawrence Rabiner and Biing-Hwang Juang, *Fundamentals of Speech Recognition*, Prentice-Hall, 1993.

[13] William M. Campbell and C. C. Broun, "A computationally scalable speaker recognition system," in *Proceedings of EUSIPCO*, 2000, pp. 457–460.

[14] Joseph P. Campbell, Jr., "Testing with the YOHO CD-ROM voice verification corpus," in *Proceedings of the International Conference on Acoustics, Speech, and Signal Processing*, 1995, pp. 341–344.

[15] Ronan Collobert and Samy Bengio, "Support vector machines for large-scale regression problems," Tech. Rep. IDIAP-RR 00-17, IDIAP, 2000.
